# Temporal Dynamics of Generalization in Neural Networks

**Changfeng Wang**
Department of Systems Engineering
University Of Pennsylvania
Philadelphia, PA 19104
fwang@pender.ee.upenn.edu

**Santosh S. Venkatesh**
Department of Electrical Engineering
University Of Pennsylvania
Philadelphia, PA 19104
venkatesh@ee.upenn.edu

## Abstract

This paper presents a rigorous characterization of how a general nonlinear learning machine generalizes during the training process when it is trained on a random sample using a gradient descent algorithm based on reduction of training error. It is shown, in particular, that best generalization performance occurs, in general, before the global minimum of the training error is achieved. The different roles played by the complexity of the machine *class* and the complexity of the *specific* machine in the class during learning are also precisely demarcated.

## 1 INTRODUCTION

In learning machines such as neural networks, two major factors that affect the 'goodness of fit' of the examples are network size (complexity) and training time. These are also the major factors that affect the generalization performance of the network.

Many theoretical studies exploring the relation between generalization performance and machine complexity support the parsimony heuristics suggested by Occam's razor, to wit that amongst machines with similar training performance one should opt for the machine of least complexity. Multitudinous numerical experiments (cf. [5]) suggest, however, that machines of larger size than strictly necessary to explain the data can yield generalization performance similar to that of smaller machines (with

similar empirical error) if learning is optimally stopped at a critical point before
the global minimum of the training error is achieved. These results seem to fly in
contradiction with a learning theoretic interpretation of Occam's razor.

In this paper, we ask the following question: How does the gradual reduction of
training error affect the generalization error when a machine of given complexity is
trained on a finite number of examples? Namely, we study the simultaneous effects
of machine size and training time on the generalization error when a finite sample
of examples is available.

Our major result is a rigorous characterization of how a given learning machine
generalizes during the training process when it is trained using a learning algorithm
based on minimization of the empirical error (or a modification of the empirical
error). In particular, we are enabled to analytically determine conditions for the
existence of a finite *optimal stopping time* in learning for achieving optimal general-
ization. We interpret the results in terms of a *time-dependent, effective machine size*
which forms the link between generalization error and machine complexity *during*
learning viewed as an evolving process in time.

Our major results are obtained by introducing new theoretical tools which allow us
to obtain finer results than would otherwise be possible by direct applications of the
uniform strong laws pioneered by Vapnik and Červonenkis (henceforth refered to as
VC-theory). The different roles played by the complexity of the machine *class* and
the complexity of the *specific* machine in the class during learning are also precisely
demarcated in our results.

Since the generalization error is defined in terms of an abstract loss function, the
results find wide applicability including but not limited to regression (square-error
loss function) and density estimation (log-likelihood loss) problems.

## 2   THE LEARNING PROBLEM

We consider the problem of learning from examples a relation between two vectors
$x$ and $y$ determined by a *fixed but unknown* probability distribution $P(x, y)$. This
model includes, in particular, the input-output relation described by

$$y = g(x, \xi),  \tag{1}$$

where $g$ is some *unknown* function of $x$ and $\xi$, which are random vectors on the same
probability space. The vector $x$ can be viewed as the input to an unknown system,
$\xi$ a random noise term (possibly dependent on $x$), and $y$ the system's output.

The hypothesis class from which the learning procedure selects a candidate function
(hypothesis) approximating $g$ is a parametric family of functions $\mathcal{H}_d = \{ f(x, \theta) :
\theta \in \Theta_d \subseteq \Re^d \}$ indexed by a subset $\Theta_d$ of $d$-dimensional Euclidean space. For
example, if $x \in \Re^m$ and $y$ is a scalar, $\mathcal{H}_d$ can be the class of functions computed
by a feedforward neural network with one hidden layer comprised of $h$ neurons and
activation function $\psi$, viz.,

$$f(x, \theta) = \psi \left( \sum_{i=1}^{h} \theta_i \psi \left( \sum_{j=1}^{m} \theta_{ij} x_j + \theta_{i0} \right) + \theta_0 \right).$$

In the above, $d = (m+2)h$ denotes the number of adjustable parameters.

The goal of learning within the hypothesis class $\mathcal{H}_d$ is to find the best approximation of the relation between $x$ and $y$ in $\mathcal{H}_d$ from a finite set of $n$ examples $\mathcal{D}_n = \{(x_1, y_1), \ldots, (x_n, y_n)\}$ drawn by independent sampling from the distribution $P(x, y)$. A learning algorithm is simply a map which, for every sample $\mathcal{D}_n$ ($n \geq 1$), produces a hypothesis in $\mathcal{H}_d$.

In practical learning situations one first selects a network of fixed structure (a fixed hypothesis class $\mathcal{H}_d$), and then determines the "best" weight vector $\theta^*$ (or equivalently, the best function $f(x, \theta^*)$ in this class) using some training algorithm. The proximity of an approximation $f(x, \theta)$ to the target function $g(x, \xi)$ at each point $x$ is measured by a *loss function* $q : (f(x, \theta), g(x, \xi)) \mapsto \Re^+$. For a given hypothesis class, the function $f(\cdot, \theta)$ is completely determined by the parameter vector $\theta$. With $g$ fixed, the loss function may be written, with a slight abuse of notation, as a map $q(x, y, \theta)$ from $\Re^m \times \Re \times \Theta$ into $\Re^+$. Examples of the forms of loss functions are the familiar square-law loss function $q(x, y, \theta) = \left(g(x, \xi) - f(x, \theta)\right)^2$ commonly used in regression and learning in neural networks, and the Kulback-Leibler distance (or relative entropy) $q(x, y, \theta) = \ln \frac{p(y|f(x, \theta))}{p(y|g(x, \xi))}$ for density estimation, where $p(y \mid i(x, \theta))$ denotes the conditional density function of $y$ given $i(x, \theta)$.

The closeness of $f(\cdot, \theta)$ to $g(\cdot)$ is measured by the expected (ensemble) loss or error

$$\mathcal{E}(\theta, d) \triangleq \int q(x, y, \theta) P(dx, dy).$$

The optimal approximation $f(\cdot, \theta^*)$ is such that $\mathcal{E}(\theta^*, d) = \min_{\theta \in \Theta_d} \mathcal{E}(\theta, d)$. In similar fashion, we define the corresponding *empirical loss* (or *training error*) by

$$\mathcal{E}_n(\theta, d) = \int q(x, y, \theta) P_n(dx, dy) = \frac{1}{n} \sum_{i=1}^{n} q(x_i, y_i, \theta).$$

where $P_n$ denotes the joint empirical distribution of input-output pairs $(x, y)$. The global minimum of the empirical error over $\Theta_d$ is denoted by $\hat{\theta}$, namely, $\hat{\theta} = \arg\min_{\theta \in \Theta} \mathcal{E}_n(\theta, d)$. An iterative algorithm for minimizing $\mathcal{E}_n(\theta, d)$ (or a modification of it) over $\Theta_d$ generates at each epoch $t$ a random vector $\theta_t \colon \mathcal{D}_n \to \Theta_d$. The quantity $\mathcal{E}(\theta_t, d) = \mathbf{E} \int q(x, \theta_t) P(dx, d\xi)$ is referred to as the *generalization error* of $\theta_t$. We are interested in the properties of the process $\{\theta_t : t = 1, 2, \ldots\}$, and the time-evolution of the sequence $\{\mathcal{E}(\theta_t, d) : t = 1, 2, \ldots\}$.

Note that each $\theta_t$ is a functional of $P_n$. When $P = P_n$, learning reduces to an optimization problem. Deviations from optimality arise intrinsically as a consequence of the discrepancy between $P_n$ and $P$. The central idea of this work is to analyze the consequence of the deviation $\Delta_n \triangleq P_n - P$ on the generalization error.

To simplify notation, we henceforth suppress $d$ and write simply $\Theta$, $\mathcal{E}(\theta)$ and $\mathcal{E}_n(\theta)$ instead of $\Theta_d$, $\mathcal{E}(\theta, d)$, and $\mathcal{E}_n(\theta, d)$, respectively.

## 2.1   Regularity Conditions

We will be interested in the local behavior of learning algorithms. Consequently, we assume that $\Theta$ is a compact set, and $\theta^*$ is the unique global minimum of $\mathcal{E}(\theta)$ on $\Theta$.

It can be argued that these assumptions are an idealization of one of the following situations:

- A global algorithm is used which is able to find the global minimum of $\mathcal{E}_n(\theta)$, and we are interested in the stage of training when $\theta_t$ has entered a region $\Theta$ where $\theta^*$ is the only global minimum of $\mathcal{E}(\theta)$;
- A local algorithm is used, and the algorithm has entered a region $\Theta$ which contains $\theta^*$ as the unique global minimum of $\mathcal{E}(\theta)$ or as a unique local minimum with which we are content.

In the sequel, we write $\partial/\partial\theta$ to denote the gradient operator with respect to the vector $\theta$, and likewise write $\partial^2/\partial\theta^2$ to denote the matrix of operators $\left[\frac{\partial^2}{\partial\theta_i\partial\theta_j}\right]_{i,j=1}^d$.

In the rest of the development we assume the following *regularity conditions*:

**A1.** The loss function $q(x,y,\cdot)$ is twice continuously differential for all $\theta \in \Theta$ and for almost all $(x,y)$;

**A2.** $P(x,y)$ has compact support;

**A3.** The optimal network $\theta^*$ is an interior point of $\Theta$;

**A4.** The matrix $\Phi(\theta^*) = \frac{\partial^2}{\partial\theta^2}\mathcal{E}(\theta^*)$ is nonsingular.

These assumptions are typically satisfied in neural network applications. We will also assume that the learning algorithm converges to the global minimum of $\mathcal{E}_n(\theta)$ over $\Theta$ (note that $\hat{\theta}$ may not be a true *global* minimum, so the assumption applies to gradient descent algorithms which converge locally). It is easy to demonstrate that for each such algorithm, there exists an algorithm which decreases the empirical error monotonically at each step of iteration. Thus, without loss of generality, we also assume that all the algorithms we consider have this monotonicity property.

## 3  GENERALIZATION DYNAMICS

### 3.1  First Phase of Learning

The quality of learning based on the minimization of the empirical error depends on the value of the quantity $\sup_\Theta |\mathcal{E}_n(\theta) - \mathcal{E}(\theta)|$. Under the above assumptions, it is shown in [3] that

$$\mathcal{E}(\theta) = \mathcal{E}_n(\theta) + O_p\left(\frac{\ln n}{\sqrt{n}}\right) \quad \text{and} \quad \mathcal{E}(\hat{\theta}) = \mathcal{E}(\theta^*) + O\left(\frac{\ln n}{n}\right).$$

Therefore, for *any* iterative algorithm for minimizing $\mathcal{E}_n(\theta)$, in the initial phase of learning the reduction of training error is essentially equivalent to the reduction of generalization error. It can be further shown that this situation persists until the estimates $\theta_t$ enter an $n^{-\delta_n}$ neighborhood of $\hat{\theta}$, where $\delta_n \to 1/2$.

The basic tool we have used in arriving at this conclusion is the VC-method. The characterization of the precise generalization properties of the machine after $\theta_t$ enters an $n^{-\delta}$ neighborhood of the limiting solution needs a more precise language than can be provided by the VC-method, and is the main content of the rest of this work.

## 3.2 Learning by Gradient Descent

In the following, we focus on generalization properties when the machine is trained using the gradient descent algorithm (Backpropagation is a Gauss-Seidel implementation of this algorithm); in particular, the adaptation is governed by the recurrence

$$\theta_{t+1} = \theta_t - \epsilon \frac{\partial}{\partial \theta} \mathcal{E}_n(\theta_t) \qquad (t \geq 0), \tag{2}$$

where the positive quantity $\epsilon$ governs the rate of learning. Learning and generalization properties for other algorithms can be studied using similar techniques.

Replace $\mathcal{E}_n$ by $\mathcal{E}$ in (2) and let $\{\,\theta_t^*, t \geq 0\,\}$ denote the generated sequence of vectors. We can show (though we will not do so here) that the weight vector $\theta_t$ is asymptotically normally distributed with expectation $\theta_t^*$ and covariance matrix with all entries of order $O(\frac{1}{n})$. It is precisely the deviation of $\theta_t$ from $\theta_t^*$ caused by the perturbation of amount $\Delta_n = P_n - P$ to the true distribution $P$ which results in interesting artifacts such as a finite optimal stopping time when the number of examples is finite.

## 3.3 The Main Equation of Generalization Dynamics

Under the regularity conditions mentioned in the last section, we can find the generalization error at each epoch of learning as an explicit function of the number of iterations, machine parameters, and the initial error. Denote by $\lambda_1 \geq \lambda_2 \geq \cdots \geq \lambda_d$ the eigenvalues of the matrix $\Phi(\theta^*)$ and suppose $T$ is the orthogonal diagonalizing matrix for $\Phi(\theta^*)$, viz., $T'\phi(\theta^*)T = \operatorname{diag}(\lambda_1, \ldots, \lambda_d)$. Set $\delta = (\delta_1, \ldots, \delta_d)' \triangleq T(\theta_0 - \theta^*)$ and for each $i$ let $\nu_i$ denote the $i$th diagonal element of the $d \times d$ matrix $T'\mathbf{E}\left\{\left(\frac{\partial}{\partial \theta} q(x, \theta^*)\right)\left(\frac{\partial}{\partial \theta} q(x, \theta^*)\right)'\right\}T$. Also let $S(\theta, \rho)$ denote the open ball of radius $\rho$ at $\theta$.

MAIN THEOREM   *Under Assumptions A1–A4, the generalization error of the machine trained according to (2) is governed by the following equation for all starting points $\theta_0 \in S(\theta^*, n^{-r})$ $(0 < r \leq \frac{1}{2})$, and uniformly for all $t \geq 0$:*

$$\mathcal{E}(\theta_t) = \mathcal{E}(\theta^*) + \frac{1}{2n} \sum_{i=1}^{d} \left\{ \frac{\nu_i}{\lambda_i} \left[1 - (1 - \epsilon\lambda_i)^t\right]^2 + \delta_i^2 \lambda_i (1 - \epsilon\lambda_i)^{2t} \right\} + O(n^{-3r}). \tag{3}$$

*If $\theta_0 \notin S(\theta^*, n^{-\frac{1}{3}})$, then the generalization dynamics is governed by the following equation valid for all $r > 0$:*

$$\mathcal{E}(\theta_t) = \mathcal{E}(\theta^*) + \frac{1}{2n} \sum_{i=1}^{d} \left\{ \frac{\nu_i}{\lambda_i} + C(t_1)(1 - \epsilon\lambda_i)^t + C_i(t_1)(1 - \epsilon\lambda_i)^{2t}\delta_i^2 \right\} + O(n^{-3r}), \tag{4}$$

*where $t_1$ is the smallest $t$ such that $\mathbf{E}\,|[I - \epsilon\Phi]^{t_1}| = An^{-r}$ for some $A > 0$, and $C(t_1)$, $C_i(t_1) \geq 0$ are constants depending on network parameters and $t_1$.*

In the special case when the data is generated by the following additive noise model

$$y = g(x) + \xi, \tag{5}$$

with $\mathbf{E}\left[\xi|x\right] = 0$, and $\mathbf{E}\left[\xi^2|x\right] = \sigma^2 = constant$, if $g(x) = f(x, \theta^*)$ and the loss function $q(x, \theta)$ is given by the square-error loss function, the above equation reduces to the following form:

$$\mathcal{E}(\theta_t) = \mathcal{E}(\theta^*) + \frac{\sigma^2}{2n}\sum_{i=1}^{d}\left\{\left[1 - (1 - \epsilon\lambda_i)^t\right]^2 + \delta_i^2\lambda_i(1 - \epsilon\lambda_i)^{2t}\right\} + O(n^{-3r}).$$

In particular, if $f(x, \theta)$ is linear in $\theta$, we obtain our previous result [4] for linear machines. The result (3) is hence a substantive extension of the earlier result to very general settings. It is noted that the extension goes beyond nonlinearity and the original additive noise data generating model—we no longer require that the 'true' model be contained in the hypothesis class.

## 3.4  Effective Complexity

Write $c_i \triangleq \nu_i/\lambda_i$. The *effective complexity* of the nonlinear machine $\theta_t$ at $t$ is defined to be

$$C(\theta^*, d, t) \triangleq \sum_{i=1}^{d} c_i\left(1 - (1 - \epsilon\lambda_i)^t\right)^2.$$

Analysis shows that the term $c_i$ indicates the level of sensitivity of output of the machine to the $i$th component of the normalized weight vector, $\theta$; $C(\theta^*, d, t)$ denotes the degree to which the approximation power of the machine is invoked by the learning process at epoch $t$. Indeed, as $t \to \infty$, $C(\theta^*, d, t) \to C_d = \sum_{i=1}^{d} c_i$, which is the complexity of the limiting machine $\hat{\theta}$ which represents the maximal fitting of examples to the machine (i.e., minimized training error). For the additive noise data generating model (5) and square-error loss function, the effective complexity becomes,

$$C(\theta^*, d, t) = \sum_{i=1}^{d}\left(1 - (1 - \epsilon\lambda_i)^t\right)^2.$$

The sum can be interpreted as the *effective number of parameters used at epoch t*. At the end of training, it becomes exactly the number of parameters of the machine.

Now write $\theta_t^* \triangleq \theta^* + (\theta_0 - \theta^*)(1 - \epsilon\lambda_i)^t$. With these definitions, (3) can be rewritten to give the following approximation error and complexity error decomposition of generalization error in the learning process:

$$\mathcal{E}(\theta_t) = \mathcal{E}(\theta_t^*) + \frac{C(\theta^*, d, t)}{2n} + O(n^{-3r}) \qquad (t \geq 0). \tag{6}$$

The first term on the right-hand-side, $\mathcal{E}(\theta_t^*)$, denotes the *approximation error at epoch t* and is the error incurred in using $\theta_t^*$ as an approximation of the 'truth.' Note that the approximation error depends on time $t$ and the initial value $\theta_0$, but not the examples. Clearly, it is the error one would obtain at epoch $t$ in minimizing the function $\mathcal{E}(\theta)$ (as opposed to $\mathcal{E}_n(\theta)$) using the same learning algorithm and starting with the same step length $\epsilon$ and initial value $\theta_0$. The second term on the right-hand-side is the *complexity error at epoch t*. This is the part of the generalization error at $t$ due to the substitution of $\mathcal{E}_n(\theta)$ for $\mathcal{E}(\theta)$.

The overfitting phenomena in learning is often intuitively attributed to the 'fitting of noise.' We see that is only partly correct: it is in fact due to the increasing use of the capacity of the machine, that the complexity penalty becomes increasingly large, this being true even when the data is clean, i.e., when $\xi \equiv 0$! Therefore, we see that (6) gives an exact trade-off of the approximation error and complexity error in the learning process.

For the case of large initial error, we see from the main theorem that the complexity error is essentially the same as that at the end of training, when the initial error is reduced to about the same order as before. The reduction of the training error leads to monotone decrease in generalization error in this case.

## 3.5 Optimal Stopping Time

We can phrase the following succinct open problem in learning in neural networks: When should learning be ideally stopped? The question was answered for linear machines which is a special form of neural networks in [4]. This section extends the result to general nonlinear machines (including neural networks) in regular cases. For this purpose, we write the generalization error in the following form:

$$\mathcal{E}(\theta_t) = \mathcal{E}(\hat{\theta}) + \phi(t) + O(n^{-3r}).$$

where

$$\phi(t) \triangleq \frac{\sigma^2}{n} \sum_{i=1}^{d} \left\{ l_i (1 - \epsilon \lambda_i)^{2t} - d_i (1 - \epsilon \lambda_i)^t \right\},$$

and $d_i$ and $l_i$ are machine parameters. The time-evolution of generalization error during the learning process is completely determined by the function $\phi(t)$.

Define $t_{\min} \in \{ \tau \geq 0 : \mathcal{E}(\theta_\tau) \leq \mathcal{E}(\theta_t)$ for all $t \geq 0 \}$, that is $t_{\min}$ denotes an epoch at which the generalization error is minimized. The smallest such number will be referred to as the *optimal stopping time* of learning. In general we have $c_i > 0$ for all $i$. In this case, it is possible to determine that there is a finite optimal stopping time. More specifically, there exists two constants $t_l$ and $t_u$ which depend on the machine parameters such that $t_l \leq t_{\min} \leq t_u$. Furthermore, it can be shown that the function $\phi(t)$ decreases monotonically for $t \leq t_l$ and increases monotonically for all $t \geq t_u$. Finally, we can relate the generalization performance when learning is optimally stopped to the best achievable performance by means of the following inequality:

$$\mathcal{E}(\theta_{t_{\min}}) \leq \mathcal{E}(\theta^*) + \frac{(1 - \kappa) C_d}{2n},$$

where $\kappa = O(n^0)$ is a constant depending on $l_i$ and $d_i$'s, and is in the interval $(0, \frac{1}{4}]$, and $C_d$ denotes, as before, the limiting value of the effective machine complexity $C(\theta^*, d, t)$ as $t \to \infty$.

In the pathological case where there exists $i$ such that $c_i = 0$, there may not exist a finite optimal stopping time. However, even in such cases, it can be shown that if $\ln(1 - \epsilon \lambda_1) / \ln(1 - \epsilon \lambda_d) < 2$, a finite optimal stopping time still exists.

## 4   CONCLUDING REMARKS

This paper describes some major results of our recent work on a rigorous characterization of the generalization process in neural network types of learning machines. In particular, we have shown that reduction of training error may not lead to improved generalization performance. Two major techniques involved are the uniform weak law (VC-theory) and differentiable statistical functionals, with the former delivering an initial estimate, and the latter giving finer results. The results shows that the complexity (e.g. VC-dimension) of a machine class does not suffice to describe the rôle of machine complexity in generalization during the learning process; the appropriate complexity notion required is a time-varying and algorithm-dependent concept of effective machine complexity.

Since results in this work contain parameters which are typically unknown, they cannot be used directly in practical situations. However, it is possible to frame criteria overcoming such difficulties. More details of the work described here and its extensions and applications can be found in [3]. The methodology adopted here is also readily adapted to study the dynamical effect of regularization on the learning process [3].

### Acknowledgements

This research was supported in part by the Air Force Office of Scientific Research under grant F49620–93–1–0120.

## References

[1] Kolmogorov, A. and V. Tihomirov (1961). $\epsilon$-entropy and $\epsilon$-capacity of sets in functional spaces. *Amer. Math. Soc. Trans. (Ser. 2)*, 17:277-364.

[2] Vapnik, V. (1982). *Estimation of Dependences Based on Empirical Data.* Springer-Verlag, New York.

[3] Wang, C. (1994). *A Theory of Generalization in Learning Machines.* Ph. D. Thesis, University of Pennsylvania.

[4] Wang, C., S. S. Venkatesh, and J. S. Judd (1993). Optimal stopping and effective machine size in learning. Proceedings of NIPS'93.

[5] Weigend, A. (1993). On overtraining and the effective number of hidden units. *Proceedings of the* 1993 *Connectionist Models Summer School.* 335-342. Ed. Mozer, M. C. et al. Hillsdale, NJ: Erlbaum Associates.
